# Hierarchical Image Probability (HIP) Models

**Clay D. Spence and Lucas Parra**
Sarnoff Corporation
CN5300
Princeton, NJ 08543-5300
{cspence, lparra}@sarnoff.com

## Abstract

We formulate a model for probability distributions on image spaces. We show that any distribution of images can be factored exactly into conditional distributions of feature vectors at one resolution (pyramid level) conditioned on the image information at lower resolutions. We would like to factor this over positions in the pyramid levels to make it tractable, but such factoring may miss long-range dependencies. To fix this, we introduce hidden class labels at each pixel in the pyramid. The result is a hierarchical mixture of conditional probabilities, similar to a hidden Markov model on a tree. The model parameters can be found with maximum likelihood estimation using the EM algorithm. We have obtained encouraging preliminary results on the problems of detecting various objects in SAR images and target recognition in optical aerial images.

## 1 Introduction

Many approaches to object recognition in images estimate $\Pr(\text{class} \mid \text{image})$. By contrast, a model of the probability distribution of images, $\Pr(\text{image})$, has many attractive features. We could use this for object recognition in the usual way by training a distribution for each object class and using Bayes' rule to get $\Pr(\text{class} \mid \text{image}) = \Pr(\text{image} \mid \text{class}) \Pr(\text{class}) / \Pr(\text{image})$. Clearly there are many other benefits of having a model of the distribution of images, since any kind of data analysis task can be approached using knowledge of the distribution of the data. For classification we could attempt to detect unusual examples and reject them, rather than trusting the classifier's output. We could also compress, interpolate, suppress noise, extend resolution, fuse multiple images, etc.

Many image analysis algorithms use probability concepts, but few treat the distribution of images. Zhu, Wu and Mumford [9] do this by computing the maximum entropy distribution given a set of statistics for some features. This seems to work well for textures but it is not clear how well it will model the appearance of more structured objects.

There are several algorithms for modeling the distributions of features extracted from the image, instead of the image itself. The Markov Random Field (*MRF*) models are an example of this line of development; see, e.g., [5, 4]. Unfortunately they tend to be very expensive computationally.

In De Bonet and Viola's flexible histogram approach [2, 1], features are extracted at multiple image scales, and the resulting feature vectors are treated as a set of independent

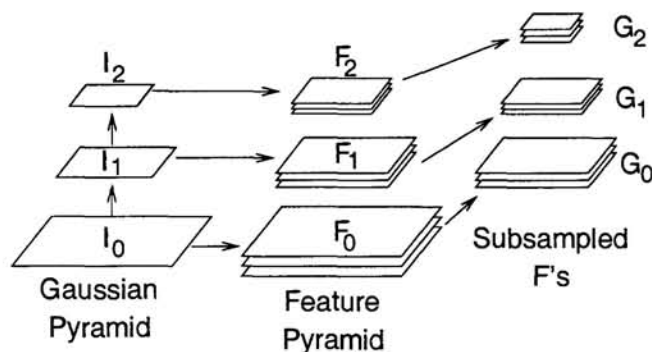

Figure 1: Pyramids and feature notation.

samples drawn from a distribution. They then model this distribution of feature vectors with Parzen windows. This has given good results, but the feature vectors from neighboring pixels are treated as independent when in fact they share exactly the same components from lower-resolutions. To fix this we might want to build a model in which the features at one pixel of one pyramid level condition the features at each of several child pixels at the next higher-resolution pyramid level. The multiscale stochastic process (*MSP*) methods do exactly that. Luettgen and Willsky [7], for example, applied a scale-space auto-regression (AR) model to texture discrimination. They use a quadtree or quadtree-like organization of the pixels in an image pyramid, and model the features in the pyramid as a stochastic process from coarse-to-fine levels along the tree. The variables in the process are hidden, and the observations are sums of these hidden variables plus noise. The Gaussian distributions are a limitation of MSP models. The result is also a model of the probability of the observations on the tree, not of the image.

All of these methods seem well-suited for modeling texture, but it is unclear how we might build the models to capture the appearance of more structured objects. We will argue below that the presence of objects in images can make local conditioning like that of the flexible histogram and MSP approaches inappropriate. In the following we present a model for probability distributions of images, in which we try to move beyond texture modeling. This hierarchical image probability (*HIP*) model is similar to a hidden Markov model on a tree, and can be learned with the EM algorithm. In preliminary tests of the model on classification tasks the performance was comparable to that of other algorithms.

## 2 Coarse-to-fine factoring of image distributions

Our goal will be to write the image distribution in a form similar to $\Pr(I) \sim \Pr(\mathbf{F}_0 \mid \mathbf{F}_1) \Pr(\mathbf{F}_1 \mid \mathbf{F}_2) \ldots$, where $\mathbf{F}_l$ is the set of feature images at pyramid level $l$. We expect that the short-range dependencies can be captured by the model's distribution of individual feature vectors, while the long-range dependencies can be captured somehow at low resolution. The large-scale structures affect finer scales by the conditioning.

In fact we can prove that a coarse-to-fine factoring like this is correct. From an image $I$ we build a Gaussian pyramid (repeatedly blur-and-subsample, with a Gaussian filter). Call the $l$-th level $I_l$, e.g., the original image is $I_0$ (Figure 1). From each Gaussian level $I_l$ we extract some set of feature images $\mathbf{F}_l$. Sub-sample these to get feature images $\mathbf{G}_l$. Note that the images in $\mathbf{G}_l$ have the same dimensions as $I_{l+1}$. We denote by $\tilde{\mathbf{G}}_l$ the set of images containing $I_{l+1}$ and the images in $\mathbf{G}_l$. We further denote the mapping from $I_l$ to $\tilde{\mathbf{G}}_l$ by $\tilde{\mathcal{G}}_l$.

Suppose now that $\tilde{\mathcal{G}}_0 : I_0 \mapsto \tilde{\mathbf{G}}_0$ is invertible. Then we can think of $\tilde{\mathcal{G}}_0$ as a change of vari-

ables. If we have a distribution on a space, its expressions in two different coordinate systems are related by multiplying by the Jacobian. In this case we get $\Pr(I_0) = |\tilde{\mathcal{G}}_0| \Pr(\tilde{G}_0)$. Since $\tilde{G}_0 = (G_0, I_1)$, we can factor $\Pr(\tilde{G}_0)$ to get $\Pr(I_0) = |\tilde{\mathcal{G}}_0| \Pr(G_0 \mid I_1) \Pr(I_1)$. If $\tilde{\mathcal{G}}_l$ is invertible for all $l \in \{0, \dots, L-1\}$ then we can simply repeat this change of variable and factoring procedure to get

$$\Pr(I) = \left[ \prod_{l=0}^{L-1} |\tilde{\mathcal{G}}_l| \Pr(G_l \mid I_{l+1}) \right] \Pr(I_L) \tag{1}$$

This is a very general result, valid for all $\Pr(I)$, no doubt with some rather mild restrictions to make the change of variables valid. The restriction that $\tilde{\mathcal{G}}_l$ be invertible is strong, but many such feature sets are known to exist, e.g., most wavelet transforms on images. We know of a few ways that this condition can be relaxed, but further work is needed here.

## 3  The need for hidden variables

For the sake of tractability we want to factor $\Pr(G_l \mid I_{l+1})$ over positions, something like $\Pr(I) \sim \prod_l \prod_{x \in I_{l+1}} \Pr\big(g_l(x) \mid f_{l+1}(x)\big)$ where $g_l(x)$ and $f_{l+1}(x)$ are the feature vectors at position $x$. The dependence of $g_l$ on $f_{l+1}$ expresses the persistence of image structures across scale, e.g., an edge is usually detectable as such in several neighboring pyramid levels. The flexible histogram and MSP methods share this structure. While it may be plausible that $f_{l+1}(x)$ has a strong influence on $g_l(x)$, we argue now that this factorization and conditioning is not enough to capture some properties of real images.

Objects in the world cause correlations and non-local dependencies in images. For example, the presence of a particular object might cause a certain kind of texture to be visible at level $l$. Usually local features $f_{l+1}$ by themselves will not contain enough information to infer the object's presence, but the entire image $I_{l+1}$ at that layer might. Thus $g_l(x)$ is influenced by more of $I_{l+1}$ than the local feature vector.

Similarly, objects create long-range dependencies. For example, an object class might result in a kind of texture across a large area of the image. If an object of this class is always present, the distribution may factor, but if such objects aren't always present and can't be inferred from lower-resolution information, the presence of the texture at one location affects the probability of its presence elsewhere.

We introduce hidden variables to represent the non-local information that is not captured by local features. They should also constrain the variability of features at the next finer scale. Denoting them collectively by $A$, we assume that conditioning on $A$ allows the distributions over feature vectors to factor. In general, the distribution over images becomes

$$\Pr(I) \propto \sum_A \left\{ \prod_{l=0}^{L-1} \prod_{x \in I_{l+1}} \Pr\big(g_l(x) \mid f_{l+1}(x), A\big) \Pr(A \mid I_L) \right\} \Pr(I_L). \tag{2}$$

As written this is absolutely general, so we need to be more specific. In particular we would like to preserve the conditioning of higher-resolution information on coarser-resolution information, and the ability to factor over positions.

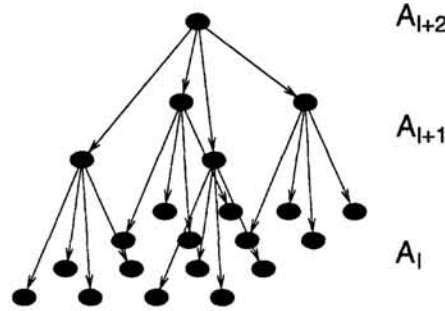

Figure 2: Tree structure of the conditional dependency between hidden variables in the HIP model. With subsampling by two, this is sometimes called a quadtree structure.

As a first model we have chosen the following structure for our HIP model:[1]

$$\Pr(I) \propto \sum_{A_0,\dots,A_{L-1}} \prod_{l=0}^{L} \prod_{x \in I_{l+1}} \left[ \Pr\big(\mathbf{g}_l(x) \mid \mathbf{f}_{l+1}(x), a_l(x)\big) \Pr\big(a_l(x) \mid a_{l+1}(x)\big) \right]. \quad (3)$$

To each position $x$ at each level $l$ we attach a hidden discrete index or label $a_l(x)$. The resulting label image $A_l$ for level $l$ has the same dimensions as the images in $\bar{\mathbf{G}}_l$.

Since $a_l(x)$ codes non-local information we can think of the labels $A_l$ as a segmentation or classification at the $l$-th pyramid level. By conditioning $a_l(x)$ on $a_{l+1}(x)$, we mean that $a_l(x)$ is conditioned on $a_{l+1}$ at the *parent* pixel of $x$. This parent-child relationship follows from the sub-sampling operation. For example, if we sub-sample by two in each direction to get $\mathbf{G}_l$ from $\mathbf{F}_l$, we condition the variable $a_l$ at $(x, y)$ in level $l$ on $a_{l+1}$ at location $(\lfloor x/2 \rfloor, \lfloor y/2 \rfloor)$ in level $l + 1$ (Figure 2). This gives the dependency graph of the hidden variables a tree structure. Such a probabilistic tree of discrete variables is sometimes referred to as a belief network. By conditioning child labels on their parents information propagates though the layers to other areas of the image while accumulating information along the way.

For the sake of simplicity we've chosen $\Pr(\mathbf{g}_l \mid \mathbf{f}_{l+1}, a_l)$ to be normal with mean $\bar{\mathbf{g}}_{l,a_l} + M_{a_l}\mathbf{f}_{l+1}$ and covariance $\Sigma_{a_l}$. We also constrain $M_{a_l}$ and $\Sigma_{a_l}$ to be diagonal.

## 4   EM algorithm

Thanks to the tree structure, the belief network for the hidden variables is relatively easy to train with an EM algorithm. The expectation step (summing over $a_l$'s) can be performed directly. If we had chosen a more densely-connected structure with each child having several parents, we would need either an approximate algorithm or Monte Carlo techniques. The expectation is weighted by the probability of a label or a parent-child pair of labels given the image. This can be computed in a fine-to-coarse-to-fine procedure, i.e. working from leaves to the root and then back out to the leaves. The method is based on belief propagation [6]. With some care an efficient algorithm can be worked out, but we omit the details due to space constraints.

Once we can compute the expectations, the normal distribution makes the M-step tractable; we simply compute the updated $\bar{\mathbf{g}}_{a_l}$, $\Sigma_{a_l}$, $M_{a_l}$, and $\Pr(a_l \mid a_{l+1})$ as combinations of various expectation values.

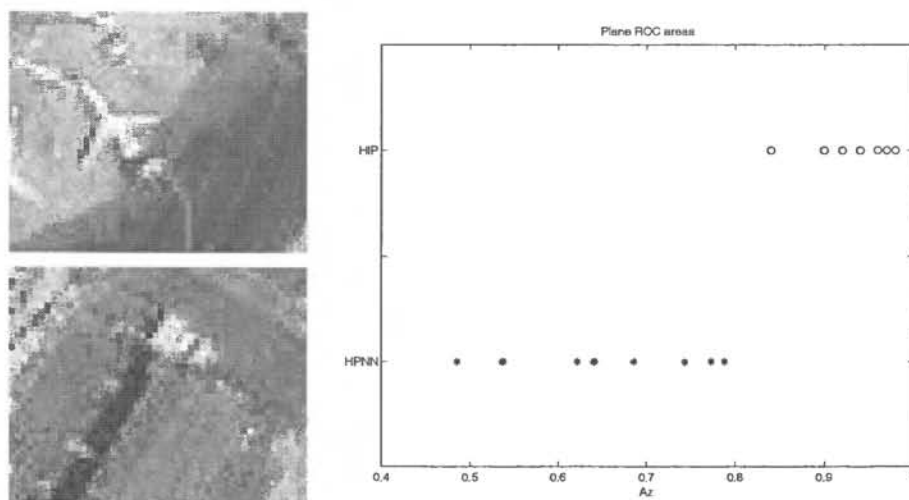

Figure 3: Examples of aircraft ROIs. On the right are $A_z$ values from a jack-knife study of detection performance of HIP and HPNN models.

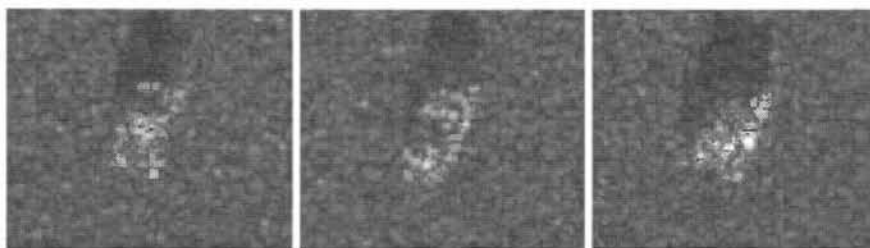

Figure 4: SAR images of three types of vehicles to be detected.

## 5  Experiments

We applied HIP to the problem of detecting aircraft in an aerial photograph of Logan airport. A simple template-matching algorithm was used to select forty candidate aircraft, twenty of which were false positives (Figure 3). Ten of the plane examples were used for training one HIP model and ten negative examples were used to train another. Because of thesmall number of examples, we performed a jack-knife study with ten random splits of the data. For features we used filter kernels that were polynomials of up to third order multiplying Gaussians. The HIP pyramid used subsampling by three in each direction. The test set ROC area for HIP had a mean of $A_z = 0.94$, while our HPNN algorithm [8] gave a mean $A_z$ of 0.65. The individual values shown in Figure 3. (We compared with the HPNN because it had given $A_z = 0.86$ on a larger set of aircraft images including these with a different set of features and subsampling by two.)

We also performed an experiment with the three target classes in the MSTAR public targets data set, to compare with the results of the flexible histogram approach of De Bonet, et al [1]. We trained three HIP models, one for each of the target vehicles BMP-2, BTR-70 and T-72 (Figure 4). As in [1] we trained each model on ten images of its class, one image for each of ten aspect angles, spaced approximately 36° apart. We trained one model for all ten images of a target, whereas De Bonet et al trained one model per image.

We first tried discriminating between vehicles of one class and other objects by thresholding $\log \Pr(I \mid \text{class})$, i.e., no model of other objects is used. For the tests, the other objects were taken from the test data for the two other vehicle classes, plus seven other vehicle classes.

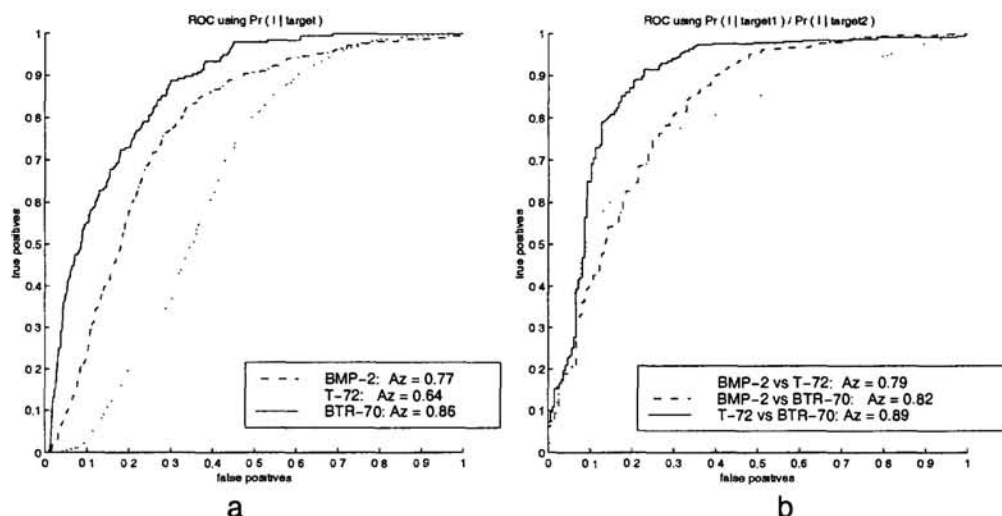

Figure 5: ROC curves for vehicle detection in SAR imagery. (a) ROC curves by thresholding HIP likelihood of desired class. (b) ROC curves for inter-class discrimination using ratios of likelihoods as given by HIP models.

There were 1,838 image from these seven other classes, 391 BMP2 test images, 196 BTR70 test images, and 386 T72 test images. The resulting ROC curves are shown in Figure 5a.

We then tried discriminating between pairs target classes using HIP model likelihood ratios, i.e., $\log \Pr(I \mid \text{class1}) - \log \Pr(I \mid \text{class2})$. Here we could not use the extra seven vehicle classes. The resulting ROC curves are shown in Figure 5b. The performance is comparable to that of the flexible histogram approach.

## 6 Conditional distributions of features

To further test the HIP model's fit to the image distribution, we computed several distributions of features $g_l(x)$ conditioned on the parent feature $f_{l+1}(x)$.[2] The empirical and computed distributions for a particular parent-child pair of features are shown in Figure 6. The conditional distributions we examined all had similar appearance, and all fit the empirical distributions well. Buccigrossi and Simoncelli [3] have reported such "bow-tie" shape conditional distributions for a variety of features. We want to point out that such conditional distributions are naturally obtained for any mixture of Gaussian distributions with varying scales and zero means. The present HIP model learns such conditionals, in effect describing the features as non-stationary Gaussian variables.

## 7 Conclusion

We have developed a class of image probability models we call hierarchical image probability or HIP models. To justify these, we showed that image distributions can be exactly represented as products over pyramid levels of distributions of sub-sampled feature images conditioned on coarser-scale image information. We argued that hidden variables are needed to capture long-range dependencies while allowing us to further factor the distributions over position. In our current model the hidden variables act as indices of mixture

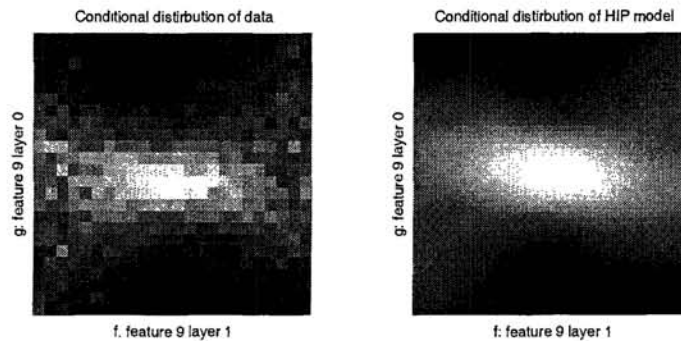

Figure 6: Empirical and HIP estimates of the distribution of a feature $g_l(x)$ conditioned on its parent feature $f_{l+1}(x)$.

components. The resulting model is somewhat like a hidden Markov model on a tree. Our early results on classification problems showed good performance.

## Acknowledgements

We thank Jeremy De Bonet and John Fisher for kindly answering questions about their work and experiments. Supported by the United States Government.

## Footnotes

[1]The proportionality factor includes $\Pr(A_L, I_L)$ which we model as $\prod_x \Pr(\mathbf{g}_L(X) \mid a_L(x)) \Pr(a_L(x))$. This is the $l = L$ factor of Equation 3, which should be read as having no quantities $\mathbf{f}_{L+1}$ or $a_{L+1}$.

[2]This is somewhat involved; $\Pr(g_l \mid f_{l+1})$ is not just $\Pr(g_l \mid f_{l+1}, a_l) \Pr(a_l)$ summed over $a_l$, but $\sum_{a_l} \Pr(g_l, a_l \mid f_{l+1}) = \sum_{a_l} \Pr(g_l \mid f_{l+1}, a_l) \Pr(a_l \mid f_{l+1})$.

## References

[1] J. S. De Bonet, P. Viola, and J. W. Fisher III. Flexible histograms: A multiresolution target discrimination model. In E. G. Zelnio, editor, *Proceedings of SPIE*, volume 3370, 1998.

[2] Jeremy S. De Bonet and Paul Viola. Texture recognition using a non-parametric multi-scale statistical model. In *Conference on Computer Vision and Pattern Recognition*. IEEE, 1998.

[3] Robert W. Buccigrossi and Eero P. Simoncelli. Image compression via joint statistical characterization in the wavelet domain. Technical Report 414, U. Penn. GRASP Laboratory, 1998. Available at ftp://ftp.cis.upenn.edu/pub/eero/buccigrossi97.ps.gz.

[4] Rama Chellappa and S. Chatterjee. Classification of textures using Gaussian Markov random fields. *IEEE Trans. ASSP*, 33:959–963, 1985.

[5] Stuart Geman and Donald Geman. Stochastic relaxation, Gibbs distributions, and the Bayesian restoration of images. *IEEE Trans. PAMI*, PAMI-6(6):194–207, November 1984.

[6] Michael I. Jordan, editor. *Learning in Graphical Models*, volume 89 of *NATO Science Series D: Behavioral and Brain Sciences*. Kluwer Academic, 1998.

[7] Mark R. Luettgen and Alan S. Willsky. Likelihood calculation for a class of multiscale stochastic models, with application to texture discrimination. *IEEE Trans. Image Proc.*, 4(2):194–207, 1995.

[8] Clay D. Spence and Paul Sajda. Applications of multi-resolution neural networks to mammography. In Michael S. Kearns, Sara A. Solla, and David A. Cohn, editors, *NIPS 11*, pages 981–988, Cambridge, MA, 1998. MIT Press.

[9] Song Chun Zhu, Ying Nian Wu, and David Mumford. Minimax entropy principle and its application to texture modeling. *Neural Computation*, 9(8):1627–1660, 1997.